# Random Features for Large-Scale Kernel Machines

**Ali Rahimi**
Intel Research Seattle
Seattle, WA 98105
ali.rahimi@intel.com

**Benjamin Recht**
Caltech IST
Pasadena, CA 91125
brecht@ist.caltech.edu

## Abstract

To accelerate the training of kernel machines, we propose to map the input data to a randomized low-dimensional feature space and then apply existing fast linear methods. The features are designed so that the inner products of the transformed data are approximately equal to those in the feature space of a user specified shift-invariant kernel. We explore two sets of random features, provide convergence bounds on their ability to approximate various radial basis kernels, and show that in large-scale classification and regression tasks linear machine learning algorithms applied to these features outperform state-of-the-art large-scale kernel machines.

## 1 Introduction

Kernel machines such as the Support Vector Machine are attractive because they can approximate any function or decision boundary arbitrarily well with enough training data. Unfortunately, methods that operate on the kernel matrix (Gram matrix) of the data scale poorly with the size of the training dataset. For example, even with the most powerful workstation, it might take days to train a nonlinear SVM on a dataset with half a million training examples. On the other hand, *linear* machines can be trained very quickly on large datasets when the dimensionality of the data is small [1, 2, 3]. One way to take advantage of these linear training algorithms for training nonlinear machines is to approximately factor the kernel matrix and to treat the columns of the factor matrix as features in a linear machine (see for example [4]). Instead, we propose to factor the kernel function itself. This factorization does not depend on the data, and allows us to convert the training and evaluation of a kernel machine into the corresponding operations of a linear machine by mapping data into a relatively low-dimensional randomized feature space. Our experiments show that these random features, combined with very simple linear learning techniques, compete favorably in speed and accuracy with state-of-the-art kernel-based classification and regression algorithms, including those that factor the kernel matrix.

The kernel trick is a simple way to generate features for algorithms that depend only on the inner product between pairs of input points. It relies on the observation that any positive definite function $k(\mathbf{x}, \mathbf{y})$ with $\mathbf{x}, \mathbf{y} \in \mathcal{R}^d$ defines an inner product and a lifting $\phi$ so that the inner product between lifted datapoints can be quickly computed as $\langle \phi(\mathbf{x}), \phi(\mathbf{y}) \rangle = k(\mathbf{x}, \mathbf{y})$. The cost of this convenience is that the algorithm accesses the data only through evaluations of $k(\mathbf{x}, \mathbf{y})$, or through the kernel matrix consisting of $k$ applied to all pairs of datapoints. As a result, large training sets incur large computational and storage costs.

Instead of relying on the implicit lifting provided by the kernel trick, we propose explicitly mapping the data to a low-dimensional Euclidean inner product space using a randomized feature map $\mathbf{z} : \mathcal{R}^d \to \mathcal{R}^D$ so that the inner product between a pair of transformed points approximates their kernel evaluation:

$$k(\mathbf{x}, \mathbf{y}) = \langle \phi(\mathbf{x}), \phi(\mathbf{y}) \rangle \approx \mathbf{z}(\mathbf{x})' \mathbf{z}(\mathbf{y}). \tag{1}$$

Unlike the kernel's lifting $\phi$, $\mathbf{z}$ is low-dimensional. Thus, we can simply transform the input with $\mathbf{z}$, and then apply fast linear learning methods to approximate the answer of the corresponding nonlinear kernel machine. In what follows, we show how to construct feature spaces that uniformly approximate popular shift-invariant kernels $k(\mathbf{x} - \mathbf{y})$ to within $\epsilon$ with only $D = O(d\epsilon^{-2} \log \frac{1}{\epsilon^2})$ dimensions, and empirically show that excellent regression and classification performance can be obtained for even smaller $D$.

In addition to giving us access to extremely fast learning algorithms, these randomized feature maps also provide a way to quickly evaluate the machine. With the kernel trick, evaluating the machine at a test point $x$ requires computing $f(\mathbf{x}) = \sum_{i=1}^{N} c_i k(\mathbf{x}_i, \mathbf{x})$, which requires $O(Nd)$ operations to compute and requires retaining much of the dataset unless the machine is very sparse. This is often unacceptable for large datasets. On the other hand, after learning a hyperplane $\mathbf{w}$, a linear machine can be evaluated by simply computing $f(x) = \mathbf{w}'\mathbf{z}(\mathbf{x})$, which, with the randomized feature maps presented here, requires only $O(D + d)$ operations and storage.

We demonstrate two randomized feature maps for approximating shift invariant kernels. Our first randomized map, presented in Section 3, consists of sinusoids randomly drawn from the Fourier transform of the kernel function we seek to approximate. Because this map is smooth, it is well-suited for interpolation tasks. Our second randomized map, presented in Section 4, partitions the input space using randomly shifted grids at randomly chosen resolutions. This mapping is not smooth, but leverages the proximity between input points, and is well-suited for approximating kernels that depend on the $L_1$ distance between datapoints. Our experiments in Section 5 demonstrate that combining these randomized maps with simple linear learning algorithms competes favorably with state-of-the-art training algorithms in a variety of regression and classification scenarios.

## 2  Related Work

The most popular methods for large-scale kernel machines are decomposition methods for solving Support Vector Machines (SVM). These methods iteratively update a subset of the kernel machine's coefficients using coordinate ascent until KKT conditions are satisfied to within a tolerance [5, 6]. While such approaches are versatile workhorses, they do not always scale to datasets with more than hundreds of thousands of datapoints for non-linear problems. To extend learning with kernel machines to these scales, several approximation schemes have been proposed for speeding up operations involving the kernel matrix.

The evaluation of the kernel function can be sped up using linear random projections [7]. Throwing away individual entries [7] or entire rows [8, 9, 10] of the kernel matrix lowers the storage and computational cost of operating on the kernel matrix. These approximations either preserve the separability of the data [8], or produce good low-rank or sparse approximations of the true kernel matrix [7, 9]. Fast multipole and multigrid methods have also been proposed for this purpose, but, while they appear to be effective on small and low-dimensional problems, they have not been demonstrated on large datasets. Further, the quality of the Hermite or Taylor approximation that these methods rely on degrades exponentially with the dimensionality of the dataset [11]. Fast nearest neighbor lookup with KD-Trees has been used to approximate multiplication with the kernel matrix, and in turn, a variety of other operations [12]. The feature map we present in Section 4 is reminiscent of KD-trees in that it partitions the input space using multi-resolution axis-aligned grids similar to those developed in [13] for embedding linear assignment problems.

## 3  Random Fourier Features

Our first set of random features project data points onto a randomly chosen line, and then pass the resulting scalar through a sinusoid (see Figure 1 and Algorithm 1). The random lines are drawn from a distribution so as to guarantee that the inner product of two transformed points approximates the desired shift-invariant kernel.

The following classical theorem from harmonic analysis provides the key insight behind this transformation:

**Theorem 1** (Bochner [15])**.** *A continuous kernel $k(x, y) = k(x - y)$ on $\mathcal{R}^d$ is positive definite if and only if $k(\delta)$ is the Fourier transform of a non-negative measure.*

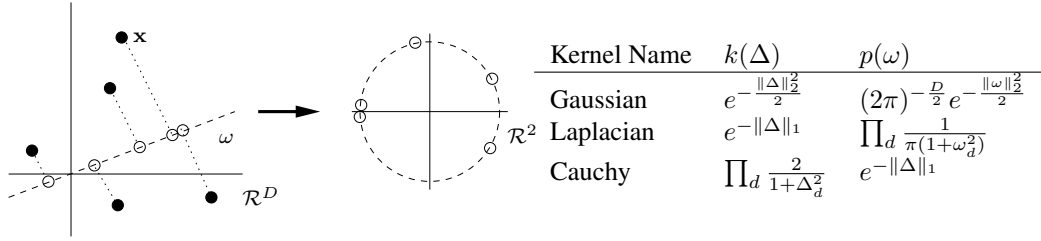

| Kernel Name | $k(\Delta)$ | $p(\omega)$ |
|---|---|---|
| Gaussian | $e^{-\frac{\|\Delta\|_2^2}{2}}$ | $(2\pi)^{-\frac{D}{2}} e^{-\frac{\|\omega\|_2^2}{2}}$ |
| Laplacian | $e^{-\|\Delta\|_1}$ | $\prod_d \frac{1}{\pi(1+\omega_d^2)}$ |
| Cauchy | $\prod_d \frac{2}{1+\Delta_d^2}$ | $e^{-\|\Delta\|_1}$ |

Figure 1: Random Fourier Features. Each component of the feature map $\mathbf{z}(\mathbf{x})$ projects $\mathbf{x}$ onto a random direction $\omega$ drawn from the Fourier transform $p(\omega)$ of $k(\Delta)$, and wraps this line onto the unit circle in $\mathcal{R}^2$. After transforming two points $\mathbf{x}$ and $\mathbf{y}$ in this way, their inner product is an unbiased estimator of $k(\mathbf{x}, \mathbf{y})$. The table lists some popular shift-invariant kernels and their Fourier transforms. To deal with non-isotropic kernels, the data may be whitened before applying one of these kernels.

If the kernel $k(\delta)$ is properly scaled, Bochner's theorem guarantees that its Fourier transform $p(\omega)$ is a proper probability distribution. Defining $\zeta_\omega(\mathbf{x}) = e^{j\omega' \mathbf{x}}$, we have

$$k(\mathbf{x} - \mathbf{y}) = \int_{\mathcal{R}^d} p(\omega) e^{j\omega'(\mathbf{x} - \mathbf{y})} \, d\omega = E_\omega[\zeta_\omega(\mathbf{x})\zeta_\omega(\mathbf{y})^*], \tag{2}$$

so $\zeta_\omega(\mathbf{x})\zeta_\omega(\mathbf{y})^*$ is an unbiased estimate of $k(\mathbf{x}, \mathbf{y})$ when $\omega$ is drawn from $p$.

To obtain a real-valued random feature for $k$, note that both the probability distribution $p(\omega)$ and the kernel $k(\Delta)$ are real, so the integrand $e^{j\omega'(\mathbf{x}-\mathbf{y})}$ may be replaced with $\cos \omega'(\mathbf{x} - \mathbf{y})$. Defining $z_\omega(\mathbf{x}) = [\cos(\mathbf{x}) \sin(\mathbf{x})]'$ gives a real-valued mapping that satisfies the condition $E[z_\omega(\mathbf{x})'z_\omega(\mathbf{y})] = k(\mathbf{x}, \mathbf{y})$, since $z_\omega(\mathbf{x})'z_\omega(\mathbf{y}) = \cos \omega'(\mathbf{x} - \mathbf{y})$. Other mappings such as $z_\omega(\mathbf{x}) = \sqrt{2}\cos(\omega'\mathbf{x} + b)$, where $\omega$ is drawn from $p(\omega)$ and $b$ is drawn uniformly from $[0, 2\pi]$, also satisfy the condition $E[z_\omega(\mathbf{x})'z_\omega(\mathbf{y})] = k(\mathbf{x}, \mathbf{y})$.

We can lower the variance of $z_\omega(\mathbf{x})'z_\omega(\mathbf{y})$ by concatenating $D$ randomly chosen $z_\omega$ into a column vector $\mathbf{z}$ and normalizing each component by $\sqrt{D}$. The inner product of points featureized by the $2D$-dimensional random feature $\mathbf{z}$, $\mathbf{z}(\mathbf{x})'\mathbf{z}(\mathbf{y}) = \frac{1}{D} \sum_{j=1}^{D} z_{\omega_j}(\mathbf{x})z_{\omega_j}(\mathbf{y})$ is a sample average of $z_{\omega_j}(\mathbf{x})z_{\omega_j}(\mathbf{y})$ and is therefore a lower variance approximation to the expectation (2).

Since $z_\omega(\mathbf{x})'z_\omega(\mathbf{y})$ is bounded between -1 and 1, for a *fixed* pair of points $\mathbf{x}$ and $\mathbf{y}$, Hoeffding's inequality guarantees exponentially fast convergence in $D$ between $\mathbf{z}(\mathbf{x})'\mathbf{z}(\mathbf{y})$ and $k(\mathbf{x}, \mathbf{y})$: $\Pr[|\mathbf{z}(\mathbf{x})'\mathbf{z}(\mathbf{y}) - k(\mathbf{x}, \mathbf{y})| \geq \epsilon] \leq 2\exp(-D\epsilon^2/2)$. Building on this observation, a much stronger assertion can be proven for every pair of points in the input space simultaneously:

**Claim 1** (Uniform convergence of Fourier features). *Let $\mathcal{M}$ be a compact subset of $\mathcal{R}^d$ with diameter* $\mathrm{diam}(\mathcal{M})$. *Then, for the mapping $\mathbf{z}$ defined in Algorithm 1, we have*

$$\Pr\left[\sup_{x,y \in \mathcal{M}} |\mathbf{z}(\mathbf{x})'\mathbf{z}(\mathbf{y}) - k(\mathbf{x}, \mathbf{y})| \geq \epsilon\right] \leq 2^8 \left(\frac{\sigma_p \, \mathrm{diam}(\mathcal{M})}{\epsilon}\right)^2 \exp\left(-\frac{D\epsilon^2}{4(d+2)}\right),$$

*where $\sigma_p^2 \equiv E_p[\omega'\omega]$ is the second moment of the Fourier transform of $k$. Further, $\sup_{x,y \in \mathcal{M}} |\mathbf{z}(\mathbf{x})'\mathbf{z}(\mathbf{y}) - k(\mathbf{y}, \mathbf{x})| \leq \epsilon$ with any constant probability when $D = \Omega\left(\frac{d}{\epsilon^2} \log \frac{\sigma_p \, \mathrm{diam}(\mathcal{M})}{\epsilon}\right)$.*

The proof of this assertion first guarantees that $\mathbf{z}(\mathbf{x})'\mathbf{z}(y)$ is close to $k(\mathbf{x} - \mathbf{y})$ for the centers of an $\epsilon$-net over $\mathcal{M} \times \mathcal{M}$. This result is then extended to the entire space using the fact that the feature map is smooth with high probability. See the Appendix for details.

By a standard Fourier identity, the scalar $\sigma_p^2$ is equal to the trace of the Hessian of $k$ at 0. It quantifies the curvature of the kernel at the origin. For the spherical Gaussian kernel, $k(\mathbf{x}, \mathbf{y}) = \exp\left(-\gamma \|\mathbf{x} - \mathbf{y}\|^2\right)$, we have $\sigma_p^2 = 2d\gamma$.

---

**Algorithm 1** Random Fourier Features.

---

**Require:** A positive definite shift-invariant kernel $k(\mathbf{x}, \mathbf{y}) = k(\mathbf{x} - \mathbf{y})$.
**Ensure:** A randomized feature map $\mathbf{z}(\mathbf{x}) : \mathcal{R}^d \to \mathcal{R}^{2D}$ so that $\mathbf{z}(\mathbf{x})'\mathbf{z}(\mathbf{y}) \approx k(\mathbf{x} - \mathbf{y})$.
  Compute the Fourier transform $p$ of the kernel $k$: $p(\omega) = \frac{1}{2\pi}\int e^{-j\omega'\Delta} k(\Delta)\, d\Delta$.
  Draw $D$ iid samples $\omega_1, \cdots, \omega_D \in \mathcal{R}^d$ from $p$.
  Let $\mathbf{z}(\mathbf{x}) \equiv \sqrt{\frac{1}{D}}\left[\cos(\omega_1'\mathbf{x}) \cdots \cos(\omega_D'\mathbf{x}) \sin(\omega_1'\mathbf{x}) \cdots \sin(\omega_D'\mathbf{x})\right]'$.

---

## 4 Random Binning Features

Our second random map partitions the input space using randomly shifted grids at randomly chosen resolutions and assigns to an input point a binary bit string that corresponds to the bin in which it falls (see Figure 2 and Algorithm 2). The grids are constructed so that the probability that two points $\mathbf{x}$ and $\mathbf{y}$ are assigned to the same bin is proportional to $k(\mathbf{x}, \mathbf{y})$. The inner product between a pair of transformed points is proportional to the number of times the two points are binned together, and is therefore an unbiased estimate of $k(\mathbf{x}, \mathbf{y})$.

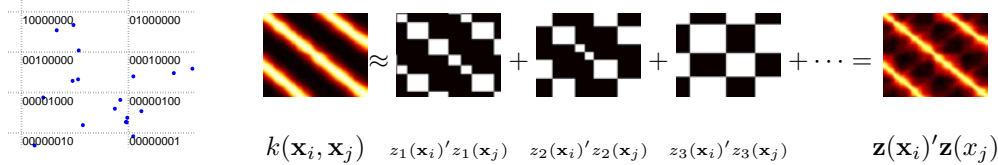

$$k(\mathbf{x}_i, \mathbf{x}_j) \quad z_1(\mathbf{x}_i)'z_1(\mathbf{x}_j) \quad z_2(\mathbf{x}_i)'z_2(\mathbf{x}_j) \quad z_3(\mathbf{x}_i)'z_3(\mathbf{x}_j) \qquad \mathbf{z}(\mathbf{x}_i)'\mathbf{z}(x_j)$$

Figure 2: Random Binning Features. (left) The algorithm repeatedly partitions the input space using a randomly shifted grid at a randomly chosen resolution and assigns to each point $\mathbf{x}$ the bit string $z(\mathbf{x})$ associated with the bin to which it is assigned. (right) The binary adjacency matrix that describes this partitioning has $z(\mathbf{x}_i)'z(\mathbf{x}_j)$ in its $ij$th entry and is an unbiased estimate of kernel matrix.

We first describe a randomized mapping to approximate the "hat" kernel $k_{hat}(x, y; \delta) = \max\left(0, 1 - \frac{|x-y|}{\delta}\right)$ on a compact subset of $\mathcal{R} \times \mathcal{R}$, then show how to construct mappings for more general separable multi-dimensional kernels. Partition the real number line with a grid of pitch $\delta$, and shift this grid randomly by an amount $u$ drawn uniformly at random from $[0, \delta]$. This grid partitions the real number line into intervals $[u + n\delta, u + (n + 1)\delta]$ for all integers $n$. The probability that two points $x$ and $y$ fall in the same bin in this grid is $\max\left(0, 1 - \frac{|x-y|}{\delta}\right)$ [13]. In other words, if we number the bins of the grid so that a point $x$ falls in bin $\hat{x} = \lfloor\frac{x-u}{\delta}\rfloor$ and $y$ falls in bin $\hat{y} = \lfloor\frac{y-u}{\delta}\rfloor$, then $\Pr_u[\hat{x} = \hat{y}|\delta] = k_{hat}(x, y; \delta)$. If we encode $\hat{x}$ as a binary indicator vector $z(x)$ over the bins, $z(x)'z(y) = 1$ if $x$ and $y$ fall in the same bin and zero otherwise, so $\Pr_u[z(x)'z(y) = 1|\delta] = E_u[z(x)'z(y)|\delta] = k_{hat}(x, y; \delta)$. Therefore $z$ is a random map for $k_{hat}$.

Now consider shift-invariant kernels that can be written as convex combinations of hat kernels on a compact subset of $\mathcal{R} \times \mathcal{R}$: $k(x, y) = \int_0^\infty k_{hat}(x, y; \delta)p(\delta)\, d\delta$. If the pitch $\delta$ of the grid is sampled from $p$, $z$ again gives a random map for $k$ because $E_{\delta,u}[z(x)'z(y)] = E_\delta\left[E_u[z(x)'z(y)|\delta]\right] = E_\delta[k_{hat}(x, y; \delta)] = k(x, y)$. That is, if the pitch $\delta$ of the grid is sampled from $p$, and the shift $u$ is drawn uniformly from $[0, \delta]$ the probability that $x$ and $y$ are binned together is $k(x, y)$. Lemma 1 in the appendix shows that $p$ can be easily recovered from $k$ by setting $p(\delta) = \delta\ddot{k}(\delta)$. For example, in the case of the Laplacian kernel, $k_{\text{Laplacian}}(x, y) = \exp(-|x - y|)$, $p(\delta)$ is the Gamma distribution $\delta\exp(-\delta)$. For the Gaussian kernel, $\ddot{k}$ is not everywhere positive, so this procedure does not yield a random map.

Random maps for separable multivariate shift-invariant kernels of the form $k(\mathbf{x} - \mathbf{y}) = \prod_{m=1}^d k_m(|x^m - y^m|)$ (such as the multivariate Laplacian kernel) can be constructed in a similar way if each $k_m$ can be written as a convex combination of hat kernels. We apply the above binning process over each dimension of $\mathcal{R}^d$ independently. The probability that $x^m$ and $y^m$ are binned together in dimension $m$ is $k_m(|x^m - y^m|)$. Since the binning process is independent across dimensions, the

probability that $\mathbf{x}$ and $\mathbf{y}$ are binned together in every dimension is $\prod_{m=1}^{d} k_m(|x^m - y^m|) = k(\mathbf{x} - \mathbf{y})$. In this multivariate case, $z(\mathbf{x})$ encodes the integer vector $[\hat{x}^1, \cdots, \hat{x}^d]$ corresponding to each bin of the $d$-dimensional grid as a binary indicator vector. In practice, to prevent overflows when computing $z(\mathbf{x})$ when $d$ is large, our implementation eliminates unoccupied bins from the representation. Since there are never more bins than training points, this ensures no overflow is possible.

We can again reduce the variance of the estimator $z(\mathbf{x})'z(\mathbf{y})$ by concatenating $P$ random binning functions $z$ into a larger list of features $\mathbf{z}$ and scaling by $\sqrt{1/P}$. The inner product $\mathbf{z}(\mathbf{x})'\mathbf{z}(\mathbf{y}) = \frac{1}{P}\sum_{p=1}^{P} z_p(\mathbf{x})'z_p(\mathbf{y})$ is the average of $P$ independent $z(\mathbf{x})'z(\mathbf{y})$ and has therefore lower variance.

Since $z(\mathbf{x})'z(\mathbf{y})$ is binary, Hoeffding's inequality guarantees that for a fixed pair of points $\mathbf{x}$ and $\mathbf{y}$, $\mathbf{z}(\mathbf{x})'\mathbf{z}(\mathbf{y})$ converges exponentially quickly to $k(\mathbf{x}, \mathbf{y})$ as a function of $P$. Again, a much stronger claim is that this convergence holds simultaneously for all points:

**Claim 2.** *Let $\mathcal{M}$ be a compact subset of $\mathcal{R}^d$ with diameter $\mathrm{diam}(\mathcal{M})$. Let $\alpha = E[1/\delta]$ and let $L_k$ denote the Lipschitz constant of $k$ with respect to the $L_1$ norm. With $\mathbf{z}$ as above, we have*

$$\Pr\left[\sup_{\mathbf{x},\mathbf{y}\in\mathcal{M}}|\mathbf{z}(\mathbf{x})'\mathbf{z}(\mathbf{y}) - k(\mathbf{x},\mathbf{y})| \le \epsilon\right] \ge 1 - 36 dP\alpha\,\mathrm{diam}(\mathcal{M})\exp\left(\frac{-\left(\frac{P\epsilon^2}{8} + \ln\frac{\epsilon}{L_k}\right)}{d+1}\right),$$

Note that $\alpha = \int_0^\infty \frac{1}{\delta}p(\delta)\,d\delta = \int_0^\infty \ddot{k}(\delta)\,d\delta$ is 1, and $L_k = 1$ for the Laplacian kernel. The proof of the claim (see the appendix) partitions $\mathcal{M} \times \mathcal{M}$ into a few small rectangular cells over which $k(\mathbf{x}, \mathbf{y})$ does not change much and $\mathbf{z}(\mathbf{x})$ and $\mathbf{z}(\mathbf{y})$ are constant. With high probability, at the centers of these cells $\mathbf{z}(\mathbf{x})'\mathbf{z}(\mathbf{y})$ is close to $k(\mathbf{x}, \mathbf{y})$, which guarantees that $k(\mathbf{x}, \mathbf{y})$ and $\mathbf{z}(\mathbf{x})'\mathbf{z}(\mathbf{y})$ are close throughout $\mathcal{M} \times \mathcal{M}$.

---

**Algorithm 2** Random Binning Features.

---

**Require:** A point $\mathbf{x} \in \mathcal{R}^d$. A kernel function $k(\mathbf{x}, \mathbf{y}) = \prod_{m=1}^{d} k_m(|x^m - y^m|)$, so that $p_m(\Delta) \equiv \Delta\ddot{k}_m(\Delta)$ is a probability distribution on $\Delta \ge 0$.
**Ensure:** A randomized feature map $\mathbf{z}(\mathbf{x})$ so that $\mathbf{z}(\mathbf{x})'\mathbf{z}(\mathbf{y}) \approx k(\mathbf{x} - \mathbf{y})$.
  **for** $p = 1 \ldots P$ **do**
      Draw grid parameters $\delta, \mathbf{u} \in \mathcal{R}^d$ with the pitch $\delta^m \sim p_m$, and shift $u^m$ from the uniform distribution on $[0, \delta^m]$.
      Let $z$ return the coordinate of the bin containing $\mathbf{x}$ as a binary indicator vector $z_p(\mathbf{x}) \equiv \mathrm{hash}(\lceil\frac{x^1-u^1}{\delta^1}\rceil, \cdots, \lceil\frac{x^d-u^d}{\delta^d}\rceil)$.
  **end for**
  $\mathbf{z}(\mathbf{x}) \equiv \sqrt{\frac{1}{P}}\left[z_1(\mathbf{x})\cdots z_P(\mathbf{x})\right]'$.

---

## 5 Experiments

The experiments summarized in Table 1 show that ridge regression with our random features is a fast way to approximate the training of supervised kernel machines. We focus our comparisons against the Core Vector Machine [14] because it was shown in [14] to be both faster and more accurate than other known approaches for training kernel machines, including, in most cases, random sampling of datapoints [8]. The experiments were conducted on the five standard large-scale datasets evaluated in [14], excluding the synthetic datasets. We replicated the results in the literature pertaining to the CVM, SVM[light], and libSVM using binaries provided by the respective authors.[1] For the random feature experiments, we trained regressors and classifiers by solving the ridge regression problem

| Dataset | Fourier+LS | Binning+LS | CVM | Exact SVM |
|---|---|---|---|---|
| CPU | 3.6% | 5.3% | 5.5% | 11% |
| regression | 20 secs | 3 mins | 51 secs | 31 secs |
| 6500 instances 21 dims | $D = 300$ | $P = 350$ | | ASVM |
| Census | 5% | 7.5% | 8.8% | 9% |
| regression | 36 secs | 19 mins | 7.5 mins | 13 mins |
| 18,000 instances 119 dims | $D = 500$ | $P = 30$ | | SVMTorch |
| Adult | 14.9% | 15.3% | 14.8% | 15.1% |
| classification | 9 secs | 1.5 mins | 73 mins | 7 mins |
| 32,000 instances 123 dims | $D = 500$ | $P = 30$ | | SVM$^{\text{light}}$ |
| Forest Cover | 11.6% | 2.2% | 2.3% | 2.2% |
| classification | 71 mins | 25 mins | 7.5 hrs | 44 hrs |
| 522,000 instances 54 dims | $D = 5000$ | $P = 50$ | | libSVM |
| KDDCUP99 (see footnote) | 7.3% | 7.3% | 6.2% (18%) | 8.3% |
| classification | 1.5 min | 35 mins | 1.4 secs (20 secs) | < 1 s |
| 4,900,000 instances 127 dims | $D = 50$ | $P = 10$ | | SVM+sampling |

Table 1: Comparison of testing error and training time between ridge regression with random features, Core Vector Machine, and various state-of-the-art exact methods reported in the literature. For classification tasks, the percent of testing points incorrectly predicted is reported, and for regression tasks, the RMS error normalized by the norm of the ground truth.

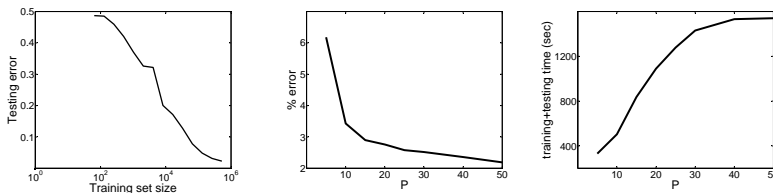

Figure 3: Accuracy on test data continues to improve as the training set grows. On the Forest dataset, using random binning, doubling the dataset size reduces testing error by up to 40% (left). Error decays quickly as $P$ grows (middle). Training time grows slowly as $P$ grows (right).

$\min_{\mathbf{w}} \|\mathbf{Z}'\mathbf{w} - \mathbf{y}\|_2^2 + \lambda\|\mathbf{w}\|_2^2$, where $\mathbf{y}$ denotes the vector of desired outputs and $\mathbf{Z}$ denotes the matrix of random features. To evaluate the resulting machine on a datapoint $\mathbf{x}$, we can simply compute $\mathbf{w}'\mathbf{z}(\mathbf{x})$. Despite its simplicity, ridge regression with random features is faster than, and provides competitive accuracy with, alternative methods. It also produces very compact functions because only $\mathbf{w}$ and a set of $O(D)$ random vectors or a hash-table of partitions need to be retained. Random Fourier features perform better on the tasks that largely rely on interpolation. On the other hand, random binning features perform better on memorization tasks (those for which the standard SVM requires many support vectors), because they explicitly preserve locality in the input space. This difference is most dramatic in the Forest dataset.

Figure 3(left) illustrates the benefit of training classifiers on larger datasets, where accuracy continues to improve as more data are used in training. Figure 3(middle) and (right) show that good performance can be obtained even from a modest number of features.

# 6 Conclusion

We have presented randomized features whose inner products uniformly approximate many popular kernels. We showed empirically that providing these features as input to a standard linear learning algorithm produces results that are competitive with state-of-the-art large-scale kernel machines in accuracy, training time, and evaluation time.

It is worth noting that hybrids of Fourier features and Binning features can be constructed by concatenating these features. While we have focused on regression and classification, our features can be applied to accelerate other kernel methods, including semi-supervised and unsupervised learning algorithms. In all of these cases, a significant computational speed-up can be achieved by first computing random features and then applying the associated linear technique.

# 7  Acknowledgements

We thank Eric Garcia for help on early versions of these features, Sameer Agarwal and James R. Lee for helpful discussions, and Erik Learned-Miller and Andres Corrada-Emmanuel for helpful corrections.

## Footnotes

[1]We include `KDDCUP99` results for completeness, but note this dataset is inherently oversampled: training an SVM (or least squares with random features) on a random sampling of 50 training examples (0.001% of the training dataset) is sufficient to consistently yield a test-error on the order of 8%. Also, while we were able to replicate the CVM's 6.2% error rate with the parameters supplied by the authors, retraining after randomly shuffling the training set results in 18% error and increases the computation time by an order of magnitude. Even on the original ordering, perturbing the CVM's regularization parameter by a mere 15% yields 49% error rate on the test set [16].

# References

[1] T. Joachims. Training linear SVMs in linear time. In *ACM Conference on Knowledge Discovery and Data Mining (KDD)*, 2006.

[2] M. C. Ferris and T. S. Munson. Interior-point methods for massive Support Vector Machines. *SIAM Journal of Optimization*, 13(3):783–804, 2003.

[3] S. Shalev-Shwartz, Y. Singer, and N. Srebro. Pegasos: Primal Estimated sub-GrAdient SOlver for SVM. In *IEEE International Conference on Machine Learning (ICML)*, 2007.

[4] D. DeCoste and D. Mazzoni. Fast query-optimized kernel machine classification via incremental approximate nearest support vectors. In *IEEE International Conference on Machine Learning (ICML)*, 2003.

[5] J. Platt. Using sparseness and analytic QP to speed training of Support Vector Machines. In *Advances in Neural Information Processing Systems (NIPS)*, 1999.

[6] C.-C. Chang and C.-J. Lin. *LIBSVM: a library for support vector machines*, 2001. Software available at http://www.csie.ntu.edu.tw/~cjlin/libsvm.

[7] D. Achlioptas, F. McSherry, and B. Schölkopf. Sampling techniques for kernel methods. In *Advances in Neural Information Processing Systems (NIPS)*, 2001.

[8] A. Blum. Random projection, margins, kernels, and feature-selection. *LNCS*, 3940:52–68, 2006.

[9] A. Frieze, R. Kannan, and S. Vempala. Fast monte-carlo algorithms for finding low-rank approximations. In *Foundations of Computer Science (FOCS)*, pages 378–390, 1998.

[10] P. Drineas and M. W. Mahoney. On the nystrom method for approximating a Gram matrix for improved kernel-based learning. In *COLT*, pages 323–337, 2005.

[11] C. Yang, R. Duraiswami, and L. Davis. Efficient kernel machines using the improved fast gauss transform. In *Advances in Neural Information Processing Systems (NIPS)*, 2004.

[12] Y. Shen, A. Y. Ng, and M. Seeger. Fast gaussian process regression using KD-Trees. In *Advances in Neural Information Processing Systems (NIPS)*, 2005.

[13] P. Indyk and N. Thaper. Fast image retrieval via embeddings. In *International Workshop on Statistical and Computational Theories of Vision*, 2003.

[14] I. W. Tsang, J. T. Kwok, and P.-M. Cheung. Core Vector Machines: Fast SVM training on very large data sets. *Journal of Machine Learning Research (JMLR)*, 6:363–392, 2005.

[15] W. Rudin. *Fourier Analysis on Groups*. Wiley Classics Library. Wiley-Interscience, New York, reprint edition edition, 1994.

[16] G. Loosli and S. Canu. Comments on the 'Core Vector Machines: Fast SVM training on very large data sets'. *Journal of Machine Learning Research (JMLR)*, 8:291–301, February 2007.

[17] F. Cucker and S. Smale. On the mathematical foundations of learning. *Bull. Amer. Soc.*, 39:1–49, 2001.

# A  Proofs

**Lemma 1.** *Suppose a function $k(\Delta) : \mathcal{R} \to \mathcal{R}$ is twice differentiable and has the form $\int_0^\infty p(\delta) \max(0, 1 - \frac{\Delta}{\delta})\, d\delta$. Then $p(\delta) = \delta \ddot{k}(\delta)$.*

*Proof.* We want $p$ so that

$$k(\Delta) = \int_0^\infty p(\delta) \max(0, 1 - \Delta/\delta)\, d\delta \tag{3}$$

$$= \int_0^\Delta p(\delta) \cdot 0\, d\delta + \int_\Delta^\infty p(\delta)(1 - \Delta/\delta)\, d\delta = \int_\Delta^\infty p(\delta)\, d\delta - \Delta \int_\Delta^\infty p(\delta)/\delta\, d\delta. \tag{4}$$

To solve for $p$, differentiate twice w.r.t. to $\Delta$ to find that $\dot{k}(\Delta) = -\int_\Delta^\infty p(\delta)/\delta\ d\delta$ and $\ddot{k}(\Delta) = p(\Delta)/\Delta$. $\qquad\qquad\square$

*Proof of Claim 1.* Define $s(\mathbf{x}, \mathbf{y}) \equiv \mathbf{z}(\mathbf{x})'\mathbf{z}(\mathbf{y})$, and $f(\mathbf{x}, \mathbf{y}) \equiv s(\mathbf{x}, \mathbf{y}) - k(\mathbf{y}, \mathbf{x})$. Since $f$, and $s$ are shift invariant, as their arguments we use $\Delta \equiv \mathbf{x} - \mathbf{y} \in \mathcal{M}_\Delta$ for notational simplicity.

$\mathcal{M}_\Delta$ is compact and has diameter at most twice $\operatorname{diam}(\mathcal{M})$, so we can find an $\epsilon$-net that covers $\mathcal{M}_\Delta$ using at most $T = (4 \operatorname{diam} \mathcal{M}/r)^d$ balls of radius $r$ [17]. Let $\{\Delta_i\}_{i=1}^T$ denote the centers of these balls, and let $L_f$ denote the Lipschitz constant of $f$. We have $|f(\Delta)| < \epsilon$ for all $\Delta \in \mathcal{M}_\Delta$ if $|f(\Delta_i)| < \epsilon/2$ and $L_f < \frac{\epsilon}{2r}$ for all $i$. We bound the probability of these two events.

Since $f$ is differentiable, $L_f = \|\nabla f(\Delta^*)\|$, where $\Delta^* = \arg\max_{\Delta \in \mathcal{M}_\Delta} \|\nabla f(\Delta)\|$. We have $E[L_f^2] = E\|\nabla f(\Delta^*)\|^2 = E\|\nabla s(\Delta^*)\|^2 - E\|\nabla k(\Delta^*)\|^2 \leq E\|\nabla s(\Delta^*)\|^2 \leq E_p\|\omega\|^2 = \sigma_p^2$, so by Markov's inequality, $\Pr[L_f^2 \geq t] \leq E[L_f^2]/t$, or

$$\Pr\left[L_f \geq \frac{\epsilon}{2r}\right] \leq \left(\frac{2r\sigma_p}{\epsilon}\right)^2. \tag{5}$$

The union bound followed by Hoeffding's inequality applied to the anchors in the $\epsilon$-net gives

$$\Pr\left[\cup_{i=1}^T |f(\Delta_i)| \geq \epsilon/8\right] \leq 2T \exp\left(-D\epsilon^2/2\right). \tag{6}$$

Combining (5) and (6) gives a bound in terms of the free variable $r$:

$$\Pr\left[\sup_{\Delta \in \mathcal{M}_\Delta} |f(\Delta)| \leq \epsilon\right] \geq 1 - 2\left(\frac{4 \operatorname{diam}(\mathcal{M})}{r}\right)^d \exp\left(-D\epsilon^2/8\right) - \left(\frac{2r\sigma_p}{\epsilon}\right)^2. \tag{7}$$

This has the form $1 - \kappa_1 r^{-d} - k_2 r^2$. Setting $r = \left(\frac{\kappa_1}{\kappa_2}\right)^{\frac{1}{d+2}}$ turns this to $1 - 2\kappa_2^{\frac{d}{d+2}} \kappa_1^{\frac{2}{d+2}}$, and assuming that $\frac{\sigma_p \operatorname{diam}(\mathcal{M})}{\epsilon} \geq 1$ and $\operatorname{diam}(\mathcal{M}) \geq 1$, proves the first part of the claim. To prove the second part of the claim, pick any probability for the RHS and solve for $D$. $\qquad\square$

*Proof of Claim 2.* $\mathcal{M}$ can be covered by rectangles over each of which $\mathbf{z}$ is constant. Let $\delta_{pm}$ be the pitch of the $p$th grid along the $m$th dimension. Each grid has at most $\lceil \operatorname{diam}(\mathcal{M})/\delta_{pm}\rceil$ bins, and $P$ overlapping grids produce at most $N_m = \sum_{g=1}^P \lceil \operatorname{diam}(\mathcal{M})/\delta_{gm}\rceil \leq \left(P + \operatorname{diam}(\mathcal{M})\sum_{p=1}^P \frac{1}{\delta_{pm}}\right)$ partitions along the $m$th dimension. The expected value of the right hand side is $P + P \operatorname{diam}(\mathcal{M})\alpha$. By Markov's inequality and the union bound, $\Pr\left[\forall_{m=1}^d N_m \leq t(P + P \operatorname{diam}(\mathcal{M})\alpha)\right] \geq 1 - d/t$. That is, with probability $1 - d/t$, along every dimension, we have at most $t(P + P \operatorname{diam}(\mathcal{M})\alpha)$ one-dimensional cells. Denote by $d_{mi}$ the width of the $i$th cell along the $m$th dimension and observe that $\sum_{i=1}^{N_m} d_{mi} \leq \operatorname{diam}(\mathcal{M})$. We further subdivide these cells into smaller rectangles of some small width $r$ to ensure that the kernel $k$ varies very little over each of these cells. This results in at most $\sum_{i=1}^{N_m} \lceil\frac{d_{mi}}{r}\rceil \leq \frac{N_m + \operatorname{diam}(\mathcal{M})}{r}$ small one-dimensional cells over each dimension. Plugging in the upper bound for $N_m$, setting $t \geq \frac{1}{\alpha P}$ and assuming $\alpha \operatorname{diam}(\mathcal{M}) \geq 1$, with probability $1 - d/t$, $\mathcal{M}$ can be covered with $T \leq \left(\frac{3tP\alpha \operatorname{diam}(\mathcal{M})}{r}\right)^d$ rectangles of side $r$ centered at $\{x_i\}_{i=1}^T$.

The condition $|z(\mathbf{x}, \mathbf{y}) - k(\mathbf{x}, \mathbf{y})| \leq \epsilon$ on $\mathcal{M} \times \mathcal{M}$ holds if $|z(\mathbf{x}_i, y_i) - k(\mathbf{x}_i, y_i)| \leq \epsilon - L_k r^d$ and $\mathbf{z}(\mathbf{x})$ is constant throughout each rectangle. With $r^d = \frac{\epsilon}{2L_k}$, the union bound followed by Hoeffding's inequality gives

$$\Pr\left[\cup_{ij}|z(\mathbf{x}_i, \mathbf{y}_j) - k(\mathbf{x}_i, \mathbf{y}_j)| \geq \epsilon/2\right] \leq 2T^2 \exp\left(-P\epsilon^2/8\right) \tag{8}$$

Combining this with the probability that $\mathbf{z}(\mathbf{x})$ is constant in each cell gives a bound in terms of $t$:

$$\Pr\left[\sup_{\mathbf{x}, \mathbf{y} \in \mathcal{M} \times \mathcal{M}} |z(\mathbf{x}, \mathbf{y}) - k(\mathbf{x}, \mathbf{y})| \leq \epsilon\right] \geq 1 - \frac{d}{t} - 2(3tP\alpha \operatorname{diam}(\mathcal{M}))^d \frac{2L_k}{\epsilon} \exp\left(-\frac{P\epsilon^2}{8}\right).$$

This has the form $1 - \kappa_1 t^{-1} - \kappa_2 t^d$. To prove the claim, set $t = \left(\frac{\kappa_1}{2\kappa_2}\right)^{\frac{1}{d+1}}$, which results in an upper bound of $1 - 3\kappa_1\kappa_2^{\frac{1}{d+1}}$. $\qquad\square$

